# Analog Cochlear Model for Multiresolution Speech Analysis

Weimin Liu,* Andreas G. Andreou and Moise H. Goldstein, Jr.
Department of Electrical and Computer Engineering
The Johns Hopkins University, Baltimore, Maryland 21218 USA

## Abstract

This paper discusses the parameterization of speech by an analog cochlear model. The tradeoff between time and frequency resolution is viewed as the fundamental difference between conventional spectrographic analysis and cochlear signal processing for broadband, rapid-changing signals. The model's response exhibits a wavelet-like analysis in the scale domain that preserves good temporal resolution; the frequency of each spectral component in a broadband signal can be accurately determined from the inter-peak intervals in the instantaneous firing rates of auditory fibers. Such properties of the cochlear model are demonstrated with natural speech and synthetic complex signals.

## 1 Introduction

As a non-parametric tool, spectrogram, or short-term Fourier transform, is widely used in analyzing non-stationary signals, such speech. Usually a window is applied to the running signal and then the Fourier transform is performed. The specific window applied determines the tradeoff between temporal and spectral resolutions of the analysis, as indicated by the uncertainty principle [1]. Since only one window is used, this tradeoff is identical for all spectral components in the signal being analyzed. This implies that conventional spectrographic signal representation and its variations are uniform resolution analysis methods. Such is also the case in parametric analysis methods, such as linear prediction coding (LPC).

In spectrographic analysis of speech, it is frequently necessary to vary the window length, or equivalently the bandwidth in order to obtain appropriate resolution in time or frequency domain. Such a practice has the effect of changing the duration-bandwidth tradeoff. Broadband (short window) analysis gives better temporal resolution to the extent that vertical voice pitch stripes can be seen; narrowband (long window) can result in better spectral resolution so that the harmonics of the pitch become apparent. A question arises: if the duty of the biological cochlea were to map a signal onto the time-frequency plane, should it be broadband or narrowband?

Neurophysiological data from the study of mammalian auditory periphery suggest that the cochlear filter is effectively broadband with regard to the harmonics in synthetic voiced speech, and a precise frequency estimation of a spectral component, such as a formant, can be determined from the analysis of the temporal patterns in the instantaneous firing rates (IFRs) of auditory nerve fibers (neurograms) [2]. A similar representation was also considered by Shamma [3].

In this paper, we will first have a close look at the spectrogram of speech signals. Then the relevant features of a cochlear model [5, 6] are described and speech processing by the model is presented illustrating good resolution in time and frequency. Careful examination of the model's output reveals that indeed it performs multiresolution analysis.

## 2   Speech Spectrogram

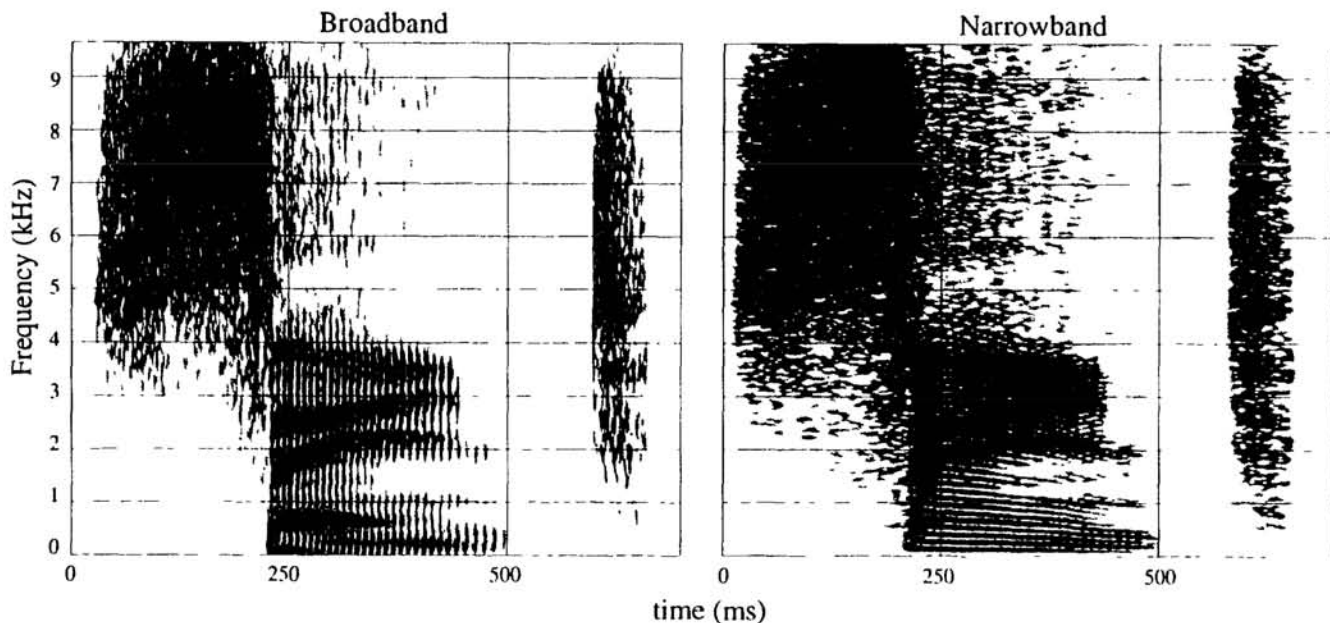

Figure 1: Broadband (6.4ms Hamming window) and narrowband (25.6ms window) spectrograms for the word "saint" spoken by a male speaker.

Figure 1 shows the broadband and narrowband spectrograms of the word "saint" spoken by a male speaker. The broadband spectrogram is usually the choice of speech analysis for several reasons. First, the fundamental frequency is considered of insignificant importance in understanding many spoken languages. Second, broadband reserves good temporal resolution, and meanwhile the representation of formants has been considered adequate. The adequacy of this notion has been seriously challenged, especially for rapidly varying events in real speech [4].

Although the vertical striation in the narrowband spectrogram indicates the pitch period, to accurately estimate the fundamental frequency $F_0$, it is often desirable to look at the narrowband spectrogram in which harmonics of $F_0$ are shown. Ideally a speech analysis method should provide multiple resolution so that both formant and harmonic information are represented simultaneously.

To further emphasize this, a synthetic signal of a tone/chirp pair was generated. The synthetic signal (Figure 2) consists of tone and chirp pairs that are separated by 100Hz. There are two 10ms gaps in both the high and low frequency tones; the chirp pair sweeps from 2900Hz-3000Hz down to 200Hz-300Hz in 100ms. The broadband spectrogram clearly shows the temporal gaps but fails to give a clear representation in frequency; the situation is reversed in the narrowband spectrogram.

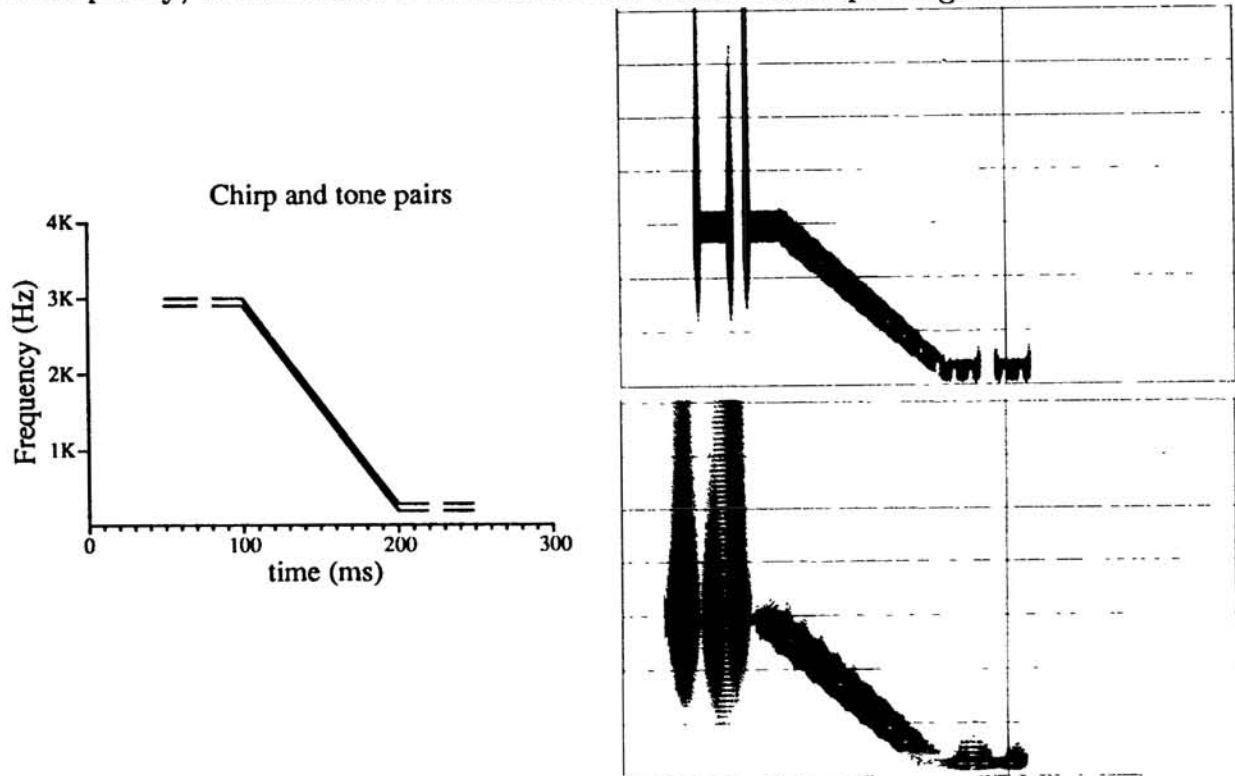

Figure 2: The synthetic tone/chirp pair and its broadband (6.4ms Hamming window) (top) and narrowband (25.6ms window) (bottom) spectrograms.

## 3   The Analog Cochlear Model

Parameterization of speech using software cochlear models has been pursued by several researchers; please refer to [5, 6] for a literature survey. The alternative to software simulations on engineering workstations, is the analog VLSI [7]. Computationally, analog VLSI models can be more effective compared to software simulations. They are also further constrained by fundamental physical limitations and scaling laws; this may direct the development of more realistic models. The constraints imposed by the technology are: power dissipation, physical extent of

computing hardware, density of interconnects, precision and noise limitations in the characteristics of the basic elements, signal dynamic range, and robust behavior and stability. Analog VLSI cochlear models have been reported by Lyon and Mead [10] at Caltech with subsequent work by Lazzaro [11] and Watts [12].

Our model [5] consists of the middle ear, the cochlear filter bank, and hair-cell/synapse modules. All the modules in the model are based on detailed biophysical and physiological studies and it builds on the software simulation and the work in our laboratory by Payton [9]. At the present time the model is implemented both as a software simulation package but also as a set of two analog VLSI chips [6] to minimize the simulation times. Even though the silicon implementation of the model is completely functional, adequate interfaces to standard engineering workstations have not been yet fully developed and therefore here we will focus on results obtained through the software simulations.

The design of the cochlear filter bank structure is the result of the effective bandwidth concept. The filter structure is flexible enough so that an appropriate set of parameters can be found to fit the neurophysiological data. In particular, the cochlear filter bank is tuned so that the model output closely resemble the auditory fibers' instantaneous firing rates (IFR) in response synthetic speech signals [2]. To do so, a fourth order section is used instead of the second order section of our earlier work [5]. Figure 3 shows the response amplitude and group delay of the filter bank that has been calibrated in this manner.

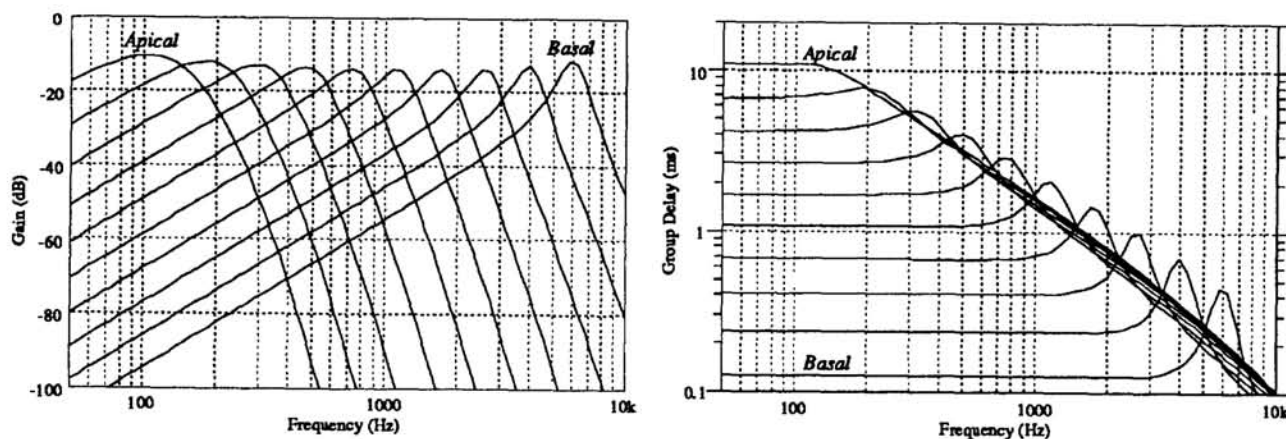

Figure 3: The amplitude and group delay of the cochlear filter bank. The curves that have higher peak frequencies in the amplitude plot and those having smaller group delays are the filter channels representing locations near the base of the cochlea.

The hair cells are the receptor cells for the hearing system. The function of hair cells and synapses in terms of signal processing is more than just rectification; besides the strong compressive nonlinearity in the mechano-electrical transduction, there are also *rapid* and *short-term* additive adaptation properties, as seen in the discharge patterns of auditory nerve fibers. Since the auditory fibers have a limited dynamic range of only 20–30dB, magnitude compression and adaptation become necessary in the transmission of acoustical signals of much wider dynamic range.

A neurotransmitter substance reservoir model, proposed by Smith and Brach-

man [8], of the hair cell and synapses that characterizes the generation of instantaneous firing rates of nerve fibers has been incorporated in the model. This is computationaly very demanding and the model benefits considerably by the analog VLSI implementation. The circuit output resembles closely the response of mammalian auditory nerve fibers [5].

## 4   Multiresolution Analysis

The conventional Fourier transform can be considered as a constant- bandwidth analysis scheme, in which the absolute frequency resolution is identical for all frequencies. A wavelet transform, on the other hand, is constant-Q in nature where the relative bandwidth is constant. The cochlear filter that is tuned to fit the experiment data is neither but is more closely related the wavelet transform, even though it required a higher Q at the base than at the apex.

The response of the cochlear model is shown in Figure 4, in the form of a neurogram. Each trace shows the IFR of a channel whose characteristic frequency is indicated on the left. The gross temporal aspects of the neurogram are rather obvious. To obtain insight into the fine time structure of IFRs, additional processing is need. One possible feature that can be extracted is the inter-peak intervals (IPI) in the IFR, which is directly related to the main spectral component in the output. The advantage of such a measure over Fourier transform is that it is not affected by the higher harmonics in the IFR. An autocorrelation and peak-picking operation were performed on the IFR output to capture the inter-peak intervals (IPIs). The procedure was similar to that by Secker-Walker and Searle [2], except that the window lengths of autocorrelation functions directly depend on the channel peak frequency. That is, for high frequency channels, shorter windows were used.

To illustrate the multiresolution nature of cochlear processing, the IPI histogram (Figure 4) across all channels are shown at each response time for the speech input "saint." Both formants and pitch frequencies are clearly shown in the composite IPI histogram.

Similarly, the cochlear model's response to the synthetic tone/chirp pairs is shown in Figure 5. The IPI histogram gives high temporal resolution for high frequencies and high spectral resolution for low frequencies such that the 10ms temporal gap in the high frequency tones and the 100Hz spacing between the two low frequency tones are precisely represented.

However in the high-frequency regions of the IPI histogram, the fact the each trace consists of a pair of tones or chirps is not clearly depicted. This limitation in the spectral resolution is the result of the relatively broad bandwidths in the high-frequency channels of the basilar membrane filter. Undoubtly the information about the 100Hz spacing in the tone/chirp pairs *is* available in the IFRs, which can be estimated from the IFR envelope which exhibits an obvious beat every 10ms (1s/100Hz) in the neurogram. Obtaining the beating information calls for a variable resolution IPI analysis scheme. For speech signals, such analysis may be necessary in pitch frequency estimation when only the IFRs of high characteristic frequencies are available.

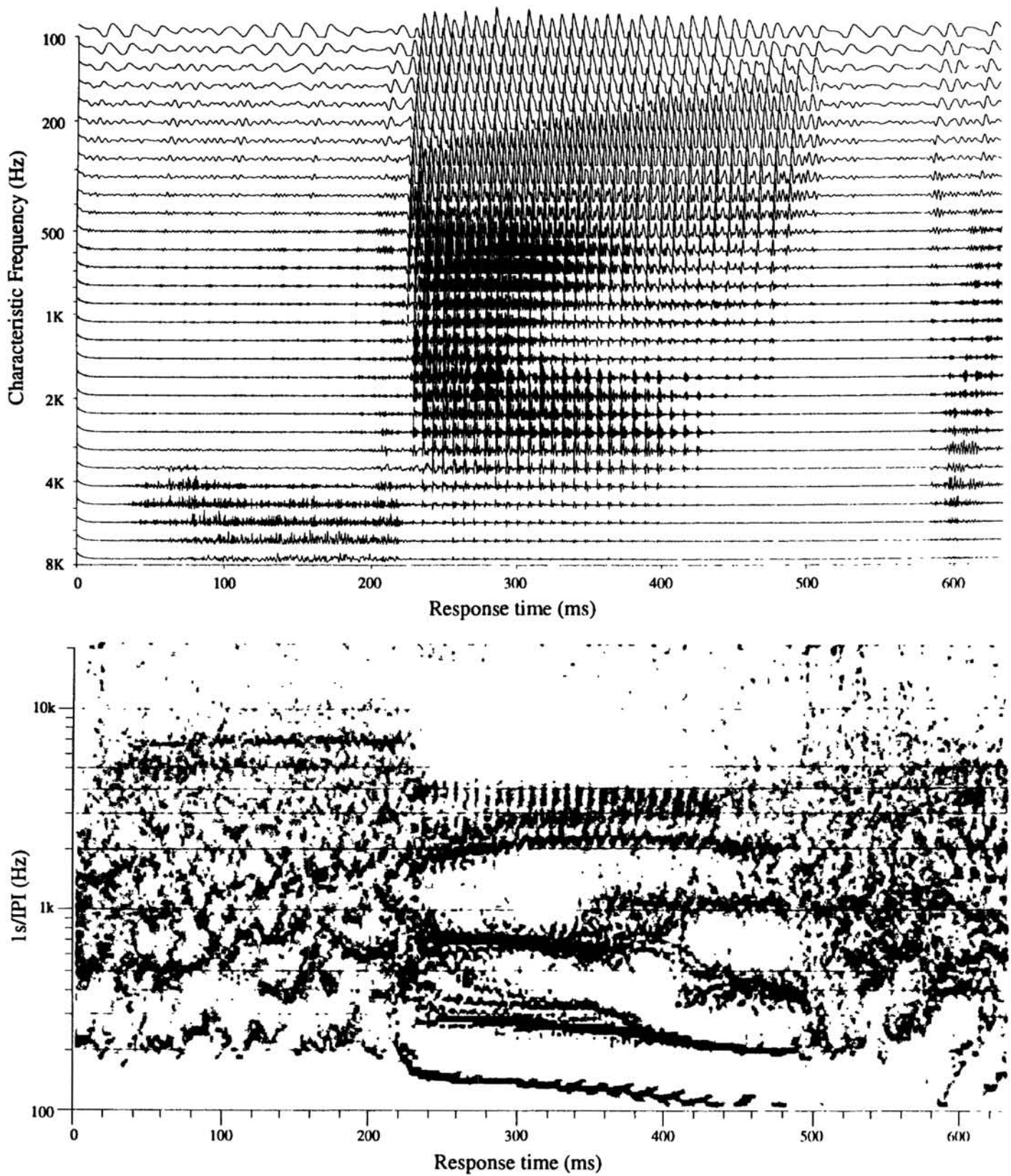

Figure 4: (*Top*) Cochlear model output, in response to "saint," in the form of neurogram. Each trace shows the IFR of one channel. Outputs from different channels are arranged according to their characteristic frequency. (*Bottom*) IPI histograms.

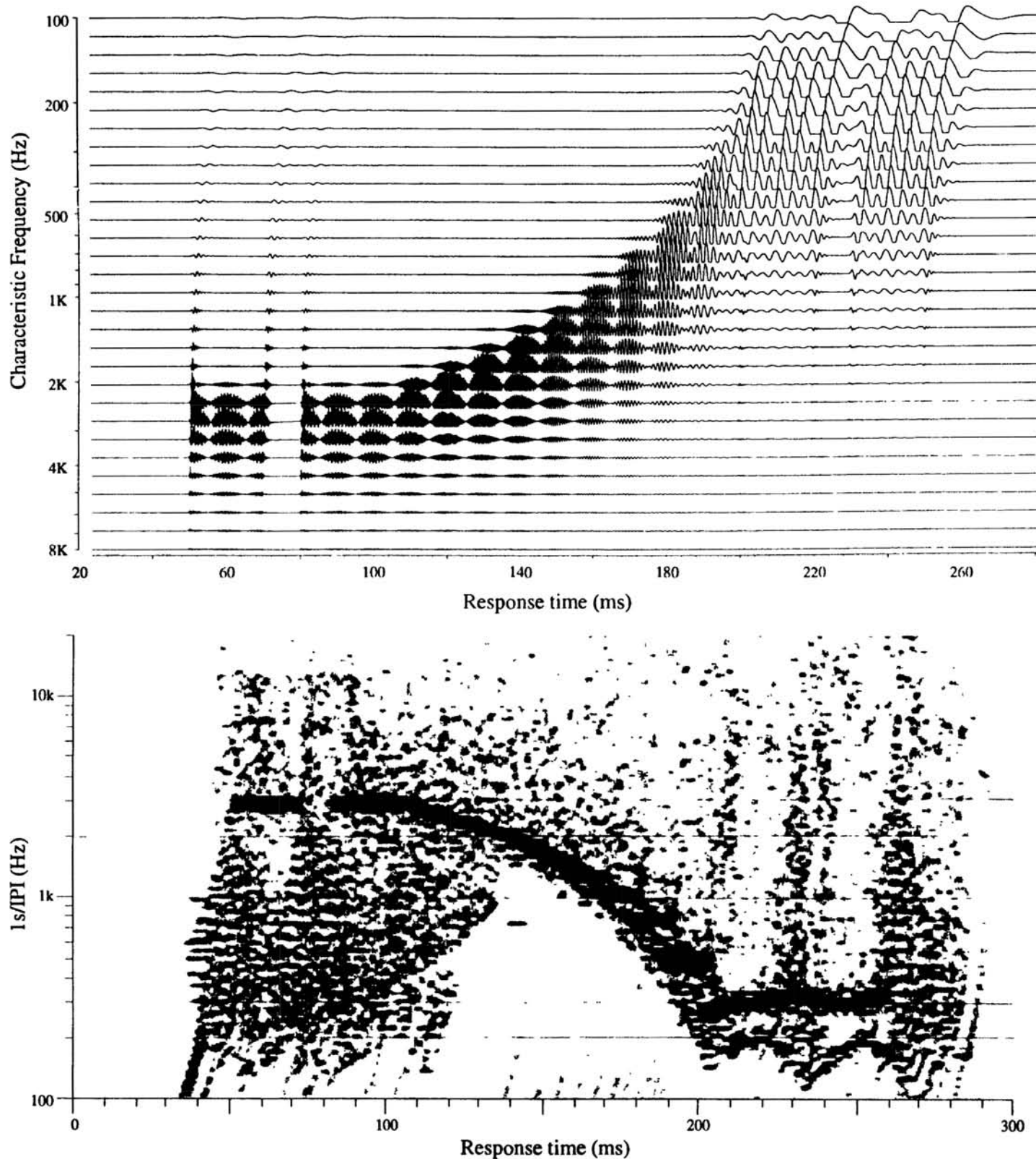

Figure 5: (*Top*) Neurogram: cochlear model response to the tone/chirp pairs. (*Bottom*) IPI histograms.

## 5   Discussion and Conclusions

We have presented an analog cochlear model that is tuned to match physiological data of mammalian cochleas in response to complex sounds and uses a small number of realistic model parameters. The response of this model has good resolution in time and in frequency, suitable for speech and broadband signal analysis. Both the analysis performed by the cochlear model and at subsequent stages (IPI) is in the *time-domain*. Processing information using temporal representations is pervasive in neural information processing systems. From an engineering perspective, it is advantageous because it results in architectures that can be efficiently implemented in analog VLSI. The cochlear model has been implemented as an analog VLSI system [6] operating in real-time. Appropriate interfaces are also being developed that will enable the silicon model to communicate with standard engineering workstations. Furthermore, refinements of the model may find applications as high-performance front-ends for various speech processing tasks.

## Footnotes

*Present address: Hughes Network Systems, Inc., 11717 Exploration Lane, Germantown, Maryland 20876 USA

## References

[1] D. Gabor. (1953) A summary of communication theory. In W. Jackson (ed.), *Communication Theory*, 1-21. London: Butterworths Scientific Pub.

[2] H.E. Secker-Walker and C.L. Searle. (1990) Time-domain analysis of auditory-nerve-fiber firing rates. *J. Acoust. Soc. Am.* 88:1427-1436.

[3] S.A. Shamma. (1985) Speech processing in the auditory system. I: representation of speech sounds in the responses of the auditory-nerve. *J. Acoust. Soc. Am.* 78:1612-1621.

[4] H.F. Siverman and Y.-T. Lee. (1987) On the spectrographic representation of rapidly time-varying speech. *Computer Speech and Language* 2:63-86.

[5] W. Liu, A.G. Andreou and M.H. Goldstein. (1992) Voiced-speech representation by an analog silicon model of the auditory periphery. *IEEE Trans. Neural Networks*, 3(3):477-487.

[6] W. Liu. (1992) *An analog cochlear model: signal representation and VLSI realization* Ph.D. Dissertation, The Johns Hopkins University.

[7] C. A. Mead, (1989) *Analog VLSI and Neural Systems*, Addison–Wesley, Reading MA.

[8] R.L. Smith and M.L. Brachman. (1982) Adaptation in auditory-nerve fibers: a revised model. *Biological Cybernetics* 44:107-120.

[9] K.L Payton. (1988) Vowel processing by a model of the auditory periphery: a comparison to eighth-nerve responses. *J. Acoust. Soc. Am.* 83:155-162.

[10] R.F. Lyon and C.A. Mead. (1988) An analog electronic cochlea. *IEEE Trans. Acoust. Speech, and Signal Process.* 36:1119-1134.

[11] J. Lazzaro and C.A. Mead. (1989) A silicon model of auditory localization. *Neural Computation* 1(1):47-57.

[12] L. Watts. (1992) *Cochlear mechanics: analysis and analog VLSI*. Ph.D. Dissertation, California Institute of Technology.
